# Incremental Learning and Selective Sampling via Parametric Optimization Framework for SVM

**Shai Fine**
IBM T. J. Watson Research Center
fshai@us.ibm.com

**Katya Scheinberg**
IBM T. J. Watson Research Center
katyas@us.ibm.com

## Abstract

We propose a framework based on a parametric quadratic programming (QP) technique to solve the support vector machine (SVM) training problem. This framework, can be specialized to obtain two SVM optimization methods. The first solves the fixed bias problem, while the second starts with an optimal solution for a fixed bias problem and adjusts the bias until the optimal value is found. The later method can be applied in conjunction with any other existing technique which obtains a fixed bias solution. Moreover, the second method can also be used independently to solve the complete SVM training problem. A combination of these two methods is more flexible than each individual method and, among other things, produces an incremental algorithm which exactly solve the *1-Norm Soft Margin* SVM optimization problem. Applying *Selective Sampling* techniques may further boost convergence.

## 1 Introduction

SVM training is a convex optimization problem which scales with the training set size rather than the input dimension. While this is usually considered to be a desired quality, in large scale problems it may cause training to be impractical. The common way to handle massive data applications is to turn to active set methods, which gradually build the set of active constraints by feeding a generic optimizer with small scale sub-problems. Active set methods guarantee to converge to the global solution, however, convergence may be very slow, it may require too many passes over the data set, and at each iteration there's an implicit computational overhead of the actual active set selection. By using some heuristics and caching mechanisms, one can, in practice, reduce this load significantly.

Another common practice is to modify the SVM optimization problem such that it wont handle the bias term directly. Instead, the bias is either fixed in advance[1] (e.g. [6]) or added as another dimension to the feature space (e.g. [4]). The advantage is that the resulting dual optimization problem does not contain the linear constraint, in which case one can suggest a procedure which updates only

one Lagrange multiplier at a time. Thus, an incremental approach, which efficiently updates an existing solution given a new training point, can be devised. Though widely used, the solution resulting from this practice has inferior generalization performances and the number of SV tends to be much higher [4].

To the best of our knowledge, the only incremental algorithm suggested so far to exactly solve the *1-Norm Soft Margin*[2] optimization problem, have been described by Cauwenberghs and Poggio at [3]. This algorithm, handles *Adiabatic* increments by solving a system of linear equations resulted from a parametric transcription of the KKT conditions. This approach is somewhat close to the one independently developed here and we offer a more thorough comparison in the discussion section.

In this paper[3] we introduce two new methods derived from parametric QP techniques. The two methods are based on the same framework, which we call Parametric Optimization for Kernel methods (POKER), and are essentially the same methodology applied to somewhat different problems. The first method solves the fixed bias problem, while the second one starts with an optimal solution for a fixed bias problem and adjusts the bias until the optimal value is found. Each of these methods can be used independently to solve the SVM training problem. The most interesting application, however, is alternating between the two methods to obtain a unique incremental algorithm. We will show how by using this approach we can adjust the optimal solution as more data becomes available, and by applying *Selective Sampling* techniques we may further boost convergence rate.

Both our methods converge after a finite number of iterations. In principle, this number may be exponential in the training set size, $n$. However, since parametric QP methods are based on the well-known Simplex method for linear programming, a similar behavior is expected: Though in theory the Simplex method is known to have exponential complexity, in practice it hardly ever displays exponential behavior. The per-iteration complexity is expected to be $O(nl)$, where $l$ is the number of active points at that iteration, with the exception of some rare cases in which the complexity is expected to be bounded by $O(nl^2)$.

## 2 Parametric QP for SVM

Any optimal solution to the *1-Norm Soft Margin* SVM optimization problem must satisfy the Karush-Kuhn-Tucker (KKT) necessary and sufficient conditions:

$$
\begin{aligned}
&\textbf{1} && \alpha_i s_i = 0, \quad i = 1, \ldots, n \\
&\textbf{2} && (c - \alpha_i)\xi_i = 0, \quad i = 1, \ldots, n \\
&\textbf{3} && y^T \alpha = 0, \\
&\textbf{4} && -Q\alpha + by + s - \xi = -e, \\
&\textbf{5} && 0 \leq \alpha \leq c, \ s \geq 0, \ \xi \geq 0.
\end{aligned}
\tag{1}
$$

where $\alpha \in \mathbf{R}^n$ is the vector of Lagrange multipliers, $b$ is the bias (scalar) and $s$ and $\xi$ are the $n$-dimensional vectors of slack and surplus variables, respectively. $y$ is a vector of labels, $\pm 1$. $Q$ is the label encoded kernel matrix, i.e. $Q_{ij} = y_i y_j K(x_i, x_j)$, $e$ is the vector of all 1's of length $n$ and $c$ is the penalty associated with errors.

If we assume that the value of the bias is fixed to some predefined value $b$, then condition **3** disappears from the system (1) and condition **4** becomes

$$-Q\alpha + s - \xi = -e - by \qquad (2)$$

Consider the following modified parametric system of KKT conditions

$$
\begin{aligned}
&\alpha_i s_i = 0, \quad i = 1, \dots, n \\
&(c - \alpha_i)\xi_i = 0, \quad i = 1, \dots, n \\
&-Q\alpha + s - \xi = p + u(-e - yb - p), \\
&0 \leq \alpha \leq c, \; s \geq 0, \; \xi \geq 0,
\end{aligned} \qquad (3)
$$

for some vector $p$. It is easy to find $p$, $\alpha$ $s$ and $\xi$ satisfying (3) for $u = 0$. For example, one may pick $\alpha = 0$, $s = e$, $\xi = 0$ and $p = -Q\alpha + s$. For $u = 1$ the systems (3) reduces to the fixed bias system. Our fixed bias method starts at a solution to (3) for $u = 0$ and by increasing $u$ while updating $\alpha$, $s$ and $\xi$ so that they satisfy (3), obtains the optimal solution for $u = 1$.

Similarly we can obtain solution to (1) by starting at a fixed bias solution and update $b$, while maintaining $\alpha$, $s$ and $\xi$ feasible for (2), until the optimal value for $b$ is reached. The optimal value of the bias is recognized when the corresponding solution satisfy (1), namely $\alpha^T y = 0$.

Both these methods are based on the same framework of adjusting a scalar parameter in the right hand side of a KKT system. In the next section we will present the method for adjusting the bias (adjusting $u$ in (3) is very similar, save for a few technical differences). An advantage of this special case is that it solves the original problem and can, in principal, be applied "from scratch".

## 3 Correcting a "Fixed Bias" Solution

Let $(\alpha(b), s(b), \xi(b))$ be a fixed bias solution for a given $b$. The algorithm that we present here is based on increasing (or decreasing) $b$ monotonically, until the optimal $b^*$ is found, while updating and maintaining $(\alpha(b), s(b), \xi(b))$.

Let us introduce some notation. For a given $b$ and and a fixed bias solution, $(\alpha(b), s(b), \xi(b))$, we partition the index set $I = \{1, \dots, n\}$ into three sets $I_0(b)$, $I_c(b)$ and $I_s(b)$ in the following way: $\forall i \in I_0(b)$ $s_i(b) > 0$ and $\alpha_i(b) = 0$, $\forall i \in I_c(b)$ $\xi_i(b) > 0$ and $\alpha_i(b) = c$ and $\forall i \in I_s(b)$ $s_i(b) = \xi_i(b) = 0$ and $0 \leq \alpha_i(b) \leq c$. It is easy to see that $I_0(b) \cup I_c(b) \cup I_s(b) = I$ and $I_0(b) \cap I_c(b) = I_c(b) \cap I_s(b) = I_0(b) \cap I_s(b) = \emptyset$. We will call the partition $(I_0(b), I_c(b), I_s(b))$ - the optimal partition for a given $b$. We will refer to $I_s$ as the *active set*. Based on partition $(I_0, I_c, I_s)$ we define $Q_{ss}$ ($Q_{cs}$ $Q_{sc}$ $Q_{cc}$, $Q_{0s}$, $Q_{00}$) as the submatrix of $Q$ whose columns are the columns of $Q$ indexed by the set $I_s$ ($I_c$, $I_s$, $I_c$, $I_0$, $I_0$) and whose rows are the rows of $Q$ indexed by $I_s$ ($I_s$, $I_c$, $I_c$, $I_s$, $I_0$). We also define $y_s$ ($y_c$, $y_0$) and $\alpha_s$ ($\alpha_c$, $\alpha_0$) and the subvectors of $y$ and $\alpha$ whose entries are indexed by $I_s$ ($I_c$, $I_0$). By $e_s$ ($e_c$) we denote a vector of all ones of the appropriate size.

Assume that we are given an initial guess[4] $b^0 < b^*$. To initiate the algorithm we

assume that we know the optimal partition $(I_0{}^0, I_c{}^0, I_s{}^0) = (I_0(b^0), I_c(b^0), I_s(b^0))$ that corresponds to $\alpha^0 = \alpha(b^0)$. We know that $\forall i \in I_0 \; \alpha_i = 0$ and $\forall i \in I_c \; \alpha_i = c$. We also know that $-Q_i\alpha + y_i b = -1, \forall i \in I_s$ (here $Q_i$ is the $i$-th row of $Q$). We can write the set of active constraints as

$$-Q_{ss}\alpha_s - cQ_{sc}e_c = -e_s - y_s b \qquad (4)$$

If $Q_{ss}$ is nonsingular (the *nondegenerate* case), then $\alpha_s$ depends linearly on scalar $b$. Similarly, we can express $s_0$ and $\xi_c$ as linear functions of $b$. If $Q_{ss}$ is singular (the *degenerate* case), then, the set of all possible solutions $\alpha_s$ changes linearly with $b$ as long as the partition remains optimal. In either case, if $0 < \alpha_s < c$, $s_0 > 0$ and $\xi_c > 0$ then sufficiently small changes in $b$ preserve these constraints. At each iteration $b$ can increase until one of the four types of inequality constraints becomes active. Then, the optimal partition is updated, new linear expressions of the active variables through $b$ are computed, and the algorithm iterates. We terminate when $y^T\alpha < 0$, that is $b > b^*$. The final iteration gives us the correct optimal active set and optimal partition; from that we can easily compute $b^*$ and $\alpha^*$.

A geometric interpretation of the algorithmic steps suggest that we are trying to move the separating hyperplane by increasing its bias and at the same time adjusting its orientation so it stays optimal for the current bias. At each iteration we move the hyperplane until either a support vector is dropped from the support set, a support vector becomes violated, a violated point becomes a support vector or an inactive point joins the support vector set.

The algorithm is guaranteed to terminate after finitely many iterations. At each iteration the algorithm covers an interval that corresponds to an optimal partition. The same partition cannot correspond to two different intervals and the number of partitions is finite, hence so is the number of iterations (cf. [1, 9]). Per-iteration complexity depends on whether an iteration is degenerate or not. A nondegenerate iteration takes $O(n|I_s|) + O(|I_s|^3)$ arithmetic operations, while a degenerate iteration should in theory take $O(n^2|I_s|^2)$ operations, but in practice it only takes[5] $O(n|I_s|^2)$. Note that the degeneracy occurs when the active support vectors are linearly dependent. The larger is the rank of the kernel matrix the less likely is such a situation. The storage requirement of the algorithm is $O(n) + O(|I_s|^2)$.

## 4  Incremental Algorithm

Incremental and on-line algorithms are aimed at training problems for which the data becomes available in the course of training. Such an algorithm, when given an optimal solution for a training set of size $n$, and additional $m$ training points, has to efficiently find the optimal solution to the extended $n + m$ training set.

Assume we have an optimal solution $(\alpha, b, s, \xi)$ for a given data set $X$ of size $n$. For each new point that is added, we take the following actions: a new Lagrange multiplier $\alpha_{n+1} = 0$ is added to the set of multipliers, then the distance to the margin is evaluated for this point. If the point is not violated, that is if $s_{n+1} = w^T x^{n+1} - y^{n+1}b - 1 > 0$, then the new positive slack $s_{n+1}$ is added to the set of slack variables. If the point is violated then $s_{n+1} = 1$ is added to the set of slack variables. (Notice, that at this point the condition $w^T x^{n+1} + y^{n+1}b + s^{n+1} = -1$ is violated.) A surplus variable $\xi_{n+1} = 0$ is also added to the set of surplus variables. The optimal partition is adjusted accordingly. The process is repeated for all the points that have to be added at the given step. If no violated points were encountered,

| | |
|---|---|
| **0** | Given data set $< X, y >$, a solution $(\alpha^0, b^0, s^0, \xi^0)$, and new points $< x, y >_{n+1}^{n+m}$ |
| **1** | Set $p = -e - by$, $\alpha_{n+i} = \xi_{n+i} = 0$, $s_{n+i} = -1 - by^{n+i} + (x^{n+i})^T w$, $i = 1, \ldots, m$ |
| **2** | If $s_{n+i} \leq 0$, Set $p_{n+i} := -(x^{n+i})^T w + 1$, $s_{n+i} = 1$ |
| | Else $p_{n+i} := -1 - by^{n+i}$ |
| **3** | $X := X \cup \{x^{n+1}, \ldots, x^{n+m}\}$, $y := (y^1, \ldots, y^n, y^{n+1}, \ldots, y^{n+m})$ |
| **4** | If $p \neq -e - b\bar{y}$ |
| | Call $\textbf{POKER}_{\textbf{fixedbias}}(X, y, \alpha, b, s, \xi, p)$ |
| | Call $\textbf{POKER}_{\textbf{adjustbias}}(X, y, \alpha, b, s, \xi)$ |
| **5** | If there are more data points go to **0**. |

Figure 1: Outline of the incremental algorithm (**AltPOKER**)

then no further action is necessary. The current solution is optimal and the bias is unchanged. If at least one point is violated, then the new set $(\alpha, b, s, \xi)$ is not feasible for the KKT system (1) with the extended data set. However, it is easy to find $p$ such that $(\alpha, b, s, \xi)$ is optimal for (3). Thus we can first apply the fixed bias algorithm to find a new solution and then apply the adjustable bias algorithm to find the optimal solution to the new extended problem (see Figure 1).

In theory adding even one point may force the algorithm to work as hard as if it were solving the problem "from scratch". But in practice it virtually never happens. In our experiments, just a few iterations of the fixed bias and adjustable bias algorithms were sufficient to find the solution to the extended problem. Overall, the computational complexity of the incremental algorithm is expected to be $O(n^2)$.

## 5 Experiments

**Convergence in Batch Mode:** The most straight-forward way to activate POKER in a batch mode is to construct the trivial partition[6] and then apply the adjustable bias algorithm to get the optimal solution. We term this method *Self-Init POKER*. Note that the initial value of the bias is most likely far away from the global solution, and as such, the results presented here should be regarded as a lower bound. We examined performances on a moderate size problem, the Abalone data set from the UCI Repository [2]. We fed the training algorithm with increasing subsets up to the whole set (of size 4177). The gender encoding (male/female/infant) was mapped into {(1,0,0),(0,1,0),(0,0,1)}. Then, the data was scaled to lie in the [-1,1] interval. We demonstrate convergence for polynomial kernel with increasing degree, which in this setting corresponds to level of difficulty. However naive our implementation is, one can observe (see Figure 2) a linear convergence rate in the batch mode.

**Convergence in Incremental Mode:** AltPOKER is the incremental algorithm described in section 4. We examined the performance on the "diabetes" problem[7] that have been used by Cauwenberghs and Poggio in [3] to test the performance of their algorithm. We demonstrate convergence for the RBF kernel with increasing penalty ("C"). Figure 3 demonstrates the advantage of the more flexible approach

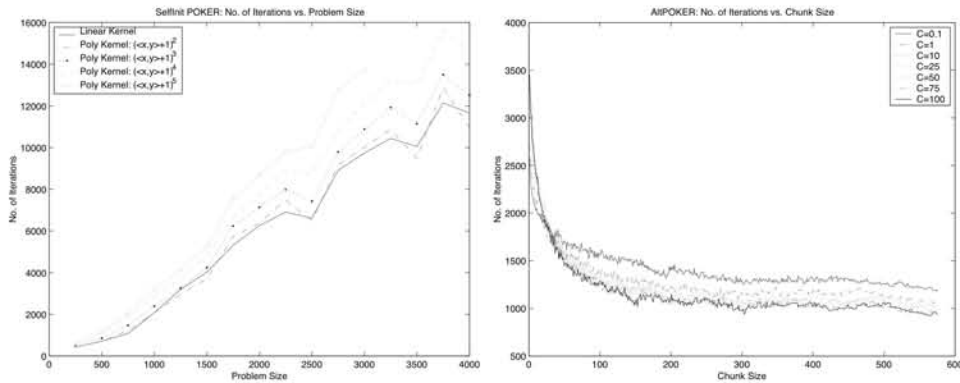

Figure 2: SelfInit POKER - Convergence in Batch mode

Figure 3: AltPOKER - Convergence in Incremental mode

which allows various increment sizes: using increments of only one point resulted in a performance of a similar scale as that of Cauwenberghs and Poggio, but with the increase of the chunk sizes we observe rapid improvement in the convergence rate.

**Selective Sampling:** We can use the incremental algorithm even in case when all the data is available in advance to improve the overall efficiency. If one can select a good representative small subset of the data set, then one can use it for training, hoping that the majority of the data points are classified correctly using the initial sampled data[8]. We applied selective sampling as a preprocess in incremental mode: At each meta-iteration, we ranked the points according to a predefined selection criterion, and then picked just the top ones for the increment.

The following selection criteria have been used in our experiments: **Cls2W** picks the closest point to the current hyperplane. This approach is inspired by active learning schemes which strive to halve the version space. However, the notion of a version space is more complex when the problem is inseparable. Thus, it is reasonable to adapt a greedy approach which selects the point that will cause the larger change in the value of the objective function.

While solving the optimization problem for all possible increments is impracticable, it may still worthwhile to approximate the potential change: **MaxSlk** picks the most violating point. This corresponds to an upper bound estimate of the change in the objective, since the value of the slack (times $c$) is an upper bound to the feasibility gap. **dObj** perform only few iterations of the adjustable bias algorithm and examine the change in the objective value. This is similar to *Strong Branching* technique which is used in branch and bound methods for integer programming. Here it provides a lower bound estimate to the change in the objective value.

Although performing only few iterations is much cheaper than converging to the optimal solution, this technique is still more demanding then previous selection methods. Hence we first ranked the points using **Cls2W (MaxSlk)** and then applied **dObj** only to the top few. Table 1 presents the application of the above mentioned criteria to three different problems. The results clearly shows that advantage of using the information obtained by **dObj** estimate.

| Selection Criteria | $\sigma^2$ | $I_s$ | $I_c$ | $I_0$ | $\sigma^2$ | $I_s$ | $I_c$ | $I_0$ | $\sigma^2$ | $I_s$ | $I_c$ | $I_0$ |
|---|---|---|---|---|---|---|---|---|---|---|---|---|
| | 400 | 4 | 11 | 9985 | 8 | 73 | 1 | 277 | 40 | 20 | 313 | 243 |
| No Selection | | | | 234 | | | | 871 | | | | 3078 |
| MaxSlk | | | | 112 | | | | 303 | | | | 3860 |
| MaxSlk+dObj | | | | 92 | | | | 269 | | | | 3184 |
| ClsW | | | | 128 | | | | 433 | | | | 2576 |
| ClsW+dObj | | | | 116 | | | | 407 | | | | 2218 |

Table 1: The impact of *Selective Sampling* on the No. of iterations of AltPOKER: Synthetic data (10Kx2), "ionosphere" [2] and "diabetes" (columns ordered resp.)

## 6 Conclusions and Discussion

We propose a new finitely convergent method that can be applied in both batch and incremental modes to solve the *1-Norm Soft Margin* SVM problem. Assuming that the number of support vectors is small compared to the size of the data, the method is expected to perform $O(n^2)$ arithmetic operations, where $n$ is the problem size. Applying *Selective Sampling* techniques may further boost convergence and reduce computation load.

Our method is independently developed, but somewhat similar to that in [3]. Our method, however, is more general - it can be applied to solve fixed bias problems as well as obtain optimal bias from a given fixed bias solution; It is not restricted to increments of size one, but rather can handle increments of arbitrary size; And, it can be used to get an estimate of the drop in the value of the objective function, which is a useful selective sampling criterion.

Finally, it is possible to extend this method to produce a true on-line algorithm, by assuming certain properties of the data. This re-introduces some very important applications of the on-line technology, such as active learning, and various forms of adaptation. Pursuing this direction with a special emphasis on massive data applications (e.g. speech related applications), is left for further study.

## Footnotes

[1]Throughout this sequel we will refer to such solution as the *fixed bias* solution.

[2]A different incremental approach stems from a geometric interpretation of the primal problem: Keerthi et al. [7] were the first to suggest a nearest point batch algorithm and Kowalczyk [8] provided the on-line version. They handled the inseparable with the well-known transformation $\tilde{W} \doteq \left(W, \sqrt{C}\xi\right)$ and $\tilde{b} \doteq b$, which establish the equivalence between the *Hard Margin* and the *2-Norm Soft Margin* optimization problems. Although the *1-Norm* and the *2-Norm* have been shown to yield equivalent generalization properties, it is often observed (cf. [7]) that the former method results in a smaller number of SV. It is obvious by the above transformation that the *1-Norm Soft Margin* is the most general SVM optimization problem.

[3]The detailed statements of the algorithms and the supporting lemmas were omitted due to space limitation, and can be found at [5].

[4] Whether $b^0 < b^*$ can be determined by evaluating $-y^T \alpha(b^0)$: if $-y^T \alpha(b^0) > 0$ then $b^0 < b^*$, otherwise $b^0 > b^*$, in which case the algorithm is essentially the same, save for obvious changes.

[5]This assumes solving such a problem by an interior point method

[6]Fixing the bias term to be large enough (positive or negative) and the Lagrange multipliers to 0 or $C$ based on their class (negative/positive) membership.

[7]available at *http://bach.ece.jhu.edu/pub/gert/svm/incremental*

[8]This is different from a full-fledged *Active Learning* scheme in which the data is not labeled, but rather queried at selected points.

## References

[1] A. B. Berkelaar, B. Jansen, K. Roos, and T. Terlaky. Sensitivity analysis in (degenerate) quadratic programming. Technical Report 96-26, Delft University, 1996.

[2] C. L. Blake and C. J Merz. UCI repository of machine learning databases, 1998.

[3] G. Cauwenberghs and T. Poggio. Incremental and decremental support vector machine learning. In *Adv. in Neural Information Processing Systems 13*, pages 409–415, 2001.

[4] N. Cristianini and J. Shawe-Taylor. *An Introductin to Support Vector Macines and Other Kernel-Based Learning Methods*. Cambridge University Press, 2000.

[5] S. Fine and K. Scheinberg. Poker: Parametric optimization framework for kernel methods. Technical report, IBM T. J. Watson Research Center, 2001. Submitted.

[6] T. T. Friess, N. Cristianini, and C. Campbell. The kernel-adaraton algorithm: A fast simple learning procedure for SVM. In *Proc. of 15th ICML*, pages 188–196, 1998.

[7] S. S. Keerthi, S. K. Shevade, C. Bhattacharyya, and K. R. K. Murthy. A fast iterative nearest point algorithm for SVM classifier design. *IEEE Trnas. NN*, 11:124–36, 2000.

[8] A. Kowalczyk. Maximal margin perceptron. In *Advances in Large Margin Classifiers*, pages 75–113. MIT Press, 2000.

[9] R. T. Rockafellar. *Conjugate Duality and Optimization*. SIAM, Philadelphia, 1974.
